# Approximately Efficient Online Mechanism Design

**David C. Parkes**
DEAS, Maxwell-Dworkin
Harvard University
parkes@eecs.harvard.edu

**Satinder Singh**
Comp. Science and Engin.
University of Michigan
baveja@umich.edu

**Dimah Yanovsky**
Harvard College
yanovsky@fas.harvard.edu

## Abstract

Online mechanism design (OMD) addresses the problem of sequential decision making in a stochastic environment with multiple self-interested agents. The goal in OMD is to make value-maximizing decisions despite this self-interest. In previous work we presented a Markov decision process (MDP)-based approach to OMD in large-scale problem domains. In practice the underlying MDP needed to solve OMD is too large and hence the mechanism must consider approximations. This raises the possibility that agents may be able to exploit the approximation for selfish gain. We adopt sparse-sampling-based MDP algorithms to implement $\epsilon$-efficient policies, and retain truth-revelation as an approximate Bayesian-Nash equilibrium. Our approach is empirically illustrated in the context of the dynamic allocation of WiFi connectivity to users in a coffeehouse.

## 1 Introduction

Mechanism design (MD) is concerned with the problem of providing incentives to implement desired system-wide outcomes in systems with multiple self-interested agents. Agents are assumed to have private information, for example about their utility for different outcomes and about their ability to implement different outcomes, and act to maximize their own utility. The MD approach to achieving multiagent coordination supposes the existence of a center that can receive messages from agents and implement an outcome and collect payments from agents. The goal of MD is to implement an outcome with desired system-wide properties in a game-theoretic equilibrium.

Classic mechanism design considers static systems in which all agents are present and a one-time decision is made about an outcome. Auctions, used in the context of resource-allocation problems, are a standard example. Online mechanism design [1] departs from this and allows agents to arrive and depart dynamically requiring decisions to be made with uncertainty about the future. Thus, an online mechanism makes a sequence of decisions without the benefit of hindsight about the valuations of the agents yet to arrive. Without the issue of incentives, the online MD problem is a classic sequential decision problem.

In prior work [6], we showed that Markov decision processes (MDPs) can be used to define an online Vickrey-Clarke-Groves (VCG) mechanism [2] that makes truth-revelation by the agents (called incentive-compatibility) a Bayesian-Nash equilibrium [5] and implements a policy that maximizes the expected summed value of all agents. This online VCG model

differs from the line of work in *online auctions*, introduced by Lavi and Nisan [4] in that it assumes that the center has a model and it handles a general decision space and any decision horizon. Computing the payments and allocations in the online VCG mechanism involves solving the MDP that defines the underlying centralized (ignoring self-interest) decision making problem. For large systems, the MDPs that need to be solved exactly become large and thus computationally infeasible.

In this paper we consider the case where the underlying centralized MDPs are indeed too large and thus must be solved approximately, as will be the case in most real applications. Of course, there are several choices of methods for solving MDPs approximately. We show that the sparse-sampling algorithm due to Kearns et al. [3] is particularly well suited to online MD because it produces the needed *local* approximations to the optimal value and action efficiently. More challengingly, regardless of our choice the agents in the system can exploit their knowledge of the mechanism's approximation algorithm to try and "cheat" the mechanism to further their own selfish interests. Our main contribution is to demonstrate that our new approximate online VCG mechanism has truth-revelation by the agents as an $\epsilon$-Bayesian-Nash equilibrium (BNE). This approximate equilibrium supposes that each agent is indifferent to within an expected utility of $\epsilon$, and will play a truthful strategy in best-response to truthful strategies of other agents if no other strategy can improve its utility by more than $\epsilon$. For any $\epsilon$, our online mechanism has a run-time that is independent of the number of states in the underlying MDP, provides an $\epsilon$-BNE, and implements a policy with expected value within $\epsilon$ of the optimal policy's value.

Our approach is empirically illustrated in the context of the dynamic allocation of WiFi connectivity to users in a coffeehouse. We demonstrate the speed-up introduced with sparse-sampling (compared with policy calculation via value-iteration), as well as the economic value of adopting an MDP-based approach over a simple fixed-price approach.

## 2  Preliminaries

Here we formalize the multiagent sequential decision problem that defines the online mechanism design (OMD) problem. The approach is centralized. Each agent is asked to report its private information (for instance about its value and its capabilities) to a central planner or mechanism upon arrival. The mechanism implements a policy based on its view of the state of the world (as reported by agents). The policy defines actions in each state, and the assumption is that all agents acquiesce to the decisions of the planner. The mechanism also collects payments from agents, which can themselves depend on the reports of agents.

**Online Mechanism Design**  We consider a finite-horizon problem with a set $T$ of time points and a sequence of decisions $k = \{k_1, \ldots, k_T\}$, where $k_t \in K_t$ and $K_t$ is the set of feasible decisions in period $t$. Agent $i \in \mathcal{I}$ arrives at time $a_i \in T$, departs at time $l_i \in T$, and has value $v_i(k) \geq 0$ for a sequence of decisions $k$. By assumption, an agent has no value for decisions outside of interval $[a_i, l_i]$. Agents also face payments, which can be collected after an agent's departure. Collectively, information $\theta_i = (a_i, l_i, v_i)$ defines the *type* of agent $i$ with $\theta_i \in \Theta$. Agent types are sampled i.i.d. from a probability distribution $f(\theta)$, assumed known to the agents and to the central mechanism. Multiple agents can arrive and depart at the same time. Agent $i$, with type $\theta_i$, receives utility $u_i(k, p; \theta_i) = v_i(k; \theta_i) - p$, for decisions $k$ and payment $p$. Agents are modeled as expected-utility maximizers.

**Definition 1** *(Online Mechanism Design) The OMD problem is to implement the sequence of decisions that maximizes the expected summed value across all agents in equilibrium, given self-interested agents with private information about valuations.*

In economic terms, an *optimal* (value-maximizing) policy is the allocatively-efficient, or simply the *efficient* policy. Note that nothing about the OMD models requires centralized

*execution* of the joint plan. Rather, the agents themselves can have capabilities to perform actions and be asked to perform particular actions by the mechanism. The agents can also have private information about the actions that they are able to perform.

**Using MDPs to Solve Online Mechanism Design.** In the MDP-based approach to solving the OMD problem the sequential decision problem is formalized as an MDP with the state at any time encapsulating both the current agent population and constraints on current decisions as reflected by decisions made previously. The reward function in the MDP is simply defined to correspond with the total reported value of all agents for all sequences of decisions.

Given types $\theta_i \in f(\theta)$ we define an MDP, $M_f$, as follows. Define the *state* of the MDP at time $t$ as the history-vector $h_t = (\theta_1, \ldots, \theta_t; k_1, \ldots, k_{t-1})$, to include the reported types up to and including period $t$ and the decisions made up to and including period $t-1$.[1] The set of all possible states at time $t$ is denoted $H_t$. The set of all possible states across all time is $H = \bigcup_{t=1}^{T+1} H_t$. The set of decisions available in state $h_t$ is $K_t(h_t)$. Given a decision $k_t \in K_t(h_t)$ in state $h_t$, there is some probability distribution $Prob(h_{t+1}|h_t, k_t)$ over possible next states $h_{t+1}$. In the setting of OMD, this probability distribution is determined by the uncertainty on new agent arrivals (as represented within $f(\theta)$), together with departures and the impact of decision $k_t$ on state.

The payoff function for the induced MDP is defined to reflect the goal of maximizing the total expected reward across all agents. In particular, payoff $R^i(h_t, k_t) = v_i(k_{\leq t}; \theta_i) - v_i(k_{\leq t-1}; \theta_i)$ becomes available from agent $i$ upon taking action $k_t$ in state $h_t$. With this, we have $\sum_{t=1}^{\tau} R^i(h_t, k_t) = v_i(k_{\leq \tau}; \theta_i)$, for all periods $\tau$ to provide the required correspondence with agent valuations. Let $R(h_t, k_t) = \sum_i R^i(h_t, k_t)$, denote the payoff obtained from all agents at time $t$. Given a (nonstationary) policy $\pi = \{\pi_1, \pi_2, \ldots, \pi_T\}$ where $\pi_t : H_t \rightarrow K_t$, an MDP defines an MDP-value function $V^\pi$ as follows: $V^\pi(h_t)$ is the expected value of the summed payoff obtained from state $h_t$ onwards under policy $\pi$, i.e., $V^\pi(h_t) = E_\pi\{R(h_t, \pi(h_t)) + R(h_{t+1}, \pi(h_{t+1})) + \cdots + R(h_T, \pi(h_T))\}$. An optimal policy $\pi^*$ is one that maximizes the MDP-value of every state in $H$.

The optimal MDP-value function $V^*$ can be computed by value-iteration, and is defined so that $V^*(h) = \max_{k \in K_t(h)}[R(h, k) + \sum_{h' \in H_{t+1}} Prob(h'|h, k)V^*(h')]$ for $t = T - 1, T - 2, \ldots, 1$ and all $h \in H_t$, with $V^*(h \in H_T) = \max_{k \in K_T(h)} R(h, k)$. Given the optimal MDP-value function, the optimal policy is derived as follows: for $t < T$, policy $\pi^*(h \in H_t)$ chooses action $\arg\max_{k \in K_t(h)}[R(h, k) + \sum_{h' \in H_{t+1}} Prob(h'|h, k)V^*(h')]$ and $\pi^*(h \in H_T) = \arg\max_{k \in K_T(h)} R(h, k)$. Let $\hat{\theta}_{\leq t'}$ denote reported types up to and including period $t'$. Let $R^i_{\leq t'}(\hat{\theta}_{\leq t'}; \pi)$ denote the total reported reward to agent $i$ up to and including period $t'$. The *commitment period* for agent $i$ is defined as the first period, $m_i$, for which $\forall t \geq m_i$, $R^i_{\leq m_i}(\hat{\theta}_{\leq m_i}; \pi) = R^i_{\leq t}(\hat{\theta}_{\leq m_i} \cup \theta'_{>m_i}; \pi)$, for any types $\theta'_{>m_i}$ still to arrive. This is the earliest period in which agent $i$'s total value is known with certainty.

Let $h_{t'}(\hat{\theta}_{\leq t'}; \pi)$ denote the state in period $t'$ given reports $\hat{\theta}_{\leq t'}$. Let $\hat{\theta}_{\leq t' \backslash i} = \hat{\theta}_{\leq t'} \backslash \hat{\theta}_i$.

**Definition 2** *(Online VCG mechanism) Given history $h \in H$, mechanism $M_{\text{vcg}} = (\Theta; \pi, p^{\text{vcg}})$ implements policy $\pi$ and collects payment,*

$$p_i^{\text{vcg}}(\hat{\theta}_{\leq m_i}; \pi) = R^i_{\leq m_i}(\hat{\theta}_{\leq m_i}; \pi) - \left[V^\pi(h_{\hat{a}_i}(\hat{\theta}_{\leq \hat{a}_i}; \pi)) - V^\pi(h_{\hat{a}_i}(\hat{\theta}_{\leq \hat{a}_i \backslash i}; \pi))\right] \quad (1)$$

*from agent $i$ in some period $t' \geq m_i$.*

Agent $i$'s payment is equal to its reported value discounted by the expected marginal value that it will contribute to the system as determined by the MDP-value function for the policy in its arrival period. The incentive-compatibility of the Online VCG mechanism requires that the MDP satisfies *stalling* which requires that the expected value from the optimal policy in every state in which an agent arrives is at least the expected value from following the optimal action in that state as though the agent had never arrived and then returning to the optimal policy. Clearly, property $K_t(h_t) \supseteq K_t(h_t \setminus \theta_i)$ in any period $t$ in which $\theta_i$ has just arrived is sufficient for stalling. Stalling is satisfied whenever an agent's arrival does not force a change in action on a policy.

**Theorem 1** *(Parkes & Singh [6]) An online VCG mechanism, $M_{\mathrm{vcg}} = (\Theta; \pi^*, p^{\mathrm{vcg}})$, based on an optimal policy $\pi^*$ for a correct MDP model that satisfies stalling is Bayesian-Nash incentive compatible and implements the optimal MDP policy.*

## 3   Solving Very Large MDPs Approximately

From Equation 1, it can be seen that making outcome and payment decisions at any point in time in an online VCG mechanism does not require a global value function or a global policy. Unlike most methods for approximately solving MDPs that compute global approximations, the sparse-sampling methods of Kearns et al. [3] compute approximate values and actions for a single state at a time. Furthermore, sparse-sampling methods provide approximation guarantees that will be important to establish equilibrium properties — they can compute an $\epsilon$-approximation to the optimal value and action in a given state in time independent of the size of the state space (though polynomial in $\frac{1}{\epsilon}$ and exponential in the time horizon). Thus, sparse-sampling methods are particularly suited to approximating online VCG and we adopt them here.

The sparse-sampling algorithm uses the MDP model $M_f$ as a generative model, i.e., as a black box from which a sample of the next-state and reward distributions for any given state-action pair can be obtained. Given a state $s$ and an approximation parameter $\epsilon$, it computes an $\epsilon$-accurate estimate of the optimal value for $s$ as follows. We make the parameterization on $\epsilon$ explicit by writing *sparse-sampling($\epsilon$)*. The algorithm builds out a depth-$T$ sampled tree in which each node is a state and each node's children are obtained by sampling each action in that state $m$ times (where $m$ is chosen to guarantee an $\epsilon$ approximation to the optimal value of $s$), and each edge is labeled with the sample reward for that transition. The algorithm computes estimates of the optimal value for nodes in the tree working backwards from the leaves as follows. The leaf-nodes have zero value. The value of a node is the maximum over the values for all actions in that node. The value of an action in a node is the summed value of the $m$ rewards on the $m$ outgoing edges for that action plus the summed value of the $m$ children of that node. The estimated optimal value of state $s$ is the value of the root node of the tree. The estimated optimal action in state $s$ is the action that leads to the largest value for the root node in the tree.

**Lemma 1** *(Adapted from Kearns, Mansour & Ng [3]) The sparse-sampling($\epsilon$) algorithm, given access to a generative model for any $n$-action MDP $M$, takes as input any state $s \in S$ and any $\epsilon > 0$, outputs an action, and satisfies the following two conditions:*

- *(Running Time) The running time of the algorithm is $O((nC)^T)$, where $C = f'(n, \frac{1}{\epsilon}, R_{\max}, T)$ and $f'$ is a polynomial function of the approximation parameter $\frac{1}{\epsilon}$, the number of actions $n$, the largest expected reward in a state $R_{\max}$ and the horizon $T$. In particular, the running time has no dependence on the number of states.*

- *(Near-Optimality) The value function of the stochastic policy implemented by the sparse-sampling($\epsilon$) algorithm, denoted $V^{ss}$, satisfies $|V^*(s) - V^{ss}(s)| \le \epsilon$ si-*

*multaneously for all states $s \in S$.*

It is straightforward to derive the following corollary from the proof of Lemma 1 in [3].

**Corollary 1** *The value function computed by the sparse-sampling($\epsilon$) algorithm, denoted $\hat{V}^{ss}$, is near-optimal in expectation, i.e., $|V^*(s) - E\{\hat{V}^{ss}(s)\}| \leq \epsilon$ simultaneously for all states $s \in S$ and where the expectation is over the randomness introduced by the sparse-sampling($\epsilon$) algorithm.*

# 4  Approximately Efficient Online Mechanism Design

In this section, we define an *approximate* online VCG mechanism and consider the effect on incentives of substituting the sparse-sampling($\epsilon$) algorithm into the online VCG mechanism. We model agents as indifferent between decisions that differ by at most a utility of $\epsilon > 0$, and consider an approximate Bayesian-Nash equilibrium. Let $v_i(\theta; \pi)$ denote the final value to agent $i$ after reports $\theta$ given policy $\pi$.

**Definition 3** *(approximate BNE) Mechanism $M_{\mathrm{vcg}} = (\Theta, \pi, p^{\mathrm{vcg}})$ is $\epsilon$-Bayesian-Nash incentive compatible if*

$$E_{\theta|\theta_{\leq t'}} \{v_i(\theta; \pi) - p_i^{\mathrm{vcg}}(\theta; \pi)\} + \epsilon \quad \geq \quad E_{\theta|\theta_{\leq t'}} \{v_i(\theta_{-i}, \hat{\theta}_i; \pi) - p_i^{\mathrm{vcg}}(\theta_{-i}, \hat{\theta}_i; \pi)\} \quad (2)$$

*where agent $i$ with type $\theta_i$ arrives in period $t'$, and with the expectation taken over future types given current reports $\theta_{\leq t'}$.*

In particular, when truth-telling is an $\epsilon$-BNE we say that the mechanism is $\epsilon$-BNE incentive compatible and no agent can improve its expected utility by more than $\epsilon > 0$, for any type, as long as other agents are bidding truthfully. Equivalently, one can interpret an $\epsilon$-BNE as an *exact* equilibrium for agents that face a computational cost of at least $\epsilon$ to compute the exact BNE.

**Definition 4** *(approximate mechanism) A sparse-sampling($\epsilon$) based approximate online VCG mechanism, $M_{\mathrm{vcg}}(\epsilon) = (\Theta; \tilde{\pi}, \tilde{p}^{\mathrm{vcg}})$, uses the sparse-sampling($\epsilon$) algorithm to implement stochastic policy $\tilde{\pi}$ and collects payment*

$$\tilde{p}_i^{\mathrm{vcg}}(\hat{\theta}_{\leq m_i}; \tilde{\pi}) \quad = \quad R_{\leq m_i}^i(\hat{\theta}_{\leq m_i}; \tilde{\pi}) - \left[\hat{V}^{ss}(h_{\hat{a}_i}(\hat{\theta}_{\leq \hat{a}_i}; \tilde{\pi})) - \hat{V}^{ss}(h_{\hat{a}_i}(\hat{\theta}_{\leq \hat{a}_i \setminus i}; \tilde{\pi}))\right]$$

*from agent $i$ in some period $t' \geq m_i$, for commitment period $m_i$.*

Our proof of incentive-compatibility first demonstrates that an approximate *delayed VCG mechanism* [1, 6] is $\epsilon$-BNE. With this, we demonstrate that the expected value of the payments in the approximate online VCG mechanism is within $3\epsilon$ of the payments in the approximate delayed VCG mechanism. The delayed VCG mechanism makes the same decisions as the online VCG mechanism, except that payments are delayed until the final period and computed as:

$$p_i^{\mathrm{Dvcg}}(\hat{\theta}; \pi) = R_{\leq T}^i(\hat{\theta}; \pi) - \left[R_{\leq T}(\hat{\theta}; \pi) - R_{\leq T}(\hat{\theta}_{-i}; \pi)\right] \qquad (3)$$

where the discount is computed ex post, once the effect of an agent on the system value is known. In an *approximate* delayed-VCG mechanism, the role of the sparse-sampling algorithm is to implement an approximate policy, as well as counterfactual policies for the worlds $\theta_{-i}$ without each agent $i$ in turn. The total reported reward to agents $\neq i$ over this counterfactual series of states is used to define the payment to agent $i$.

**Lemma 2** *Truthful bidding is an $\epsilon$-Bayesian-Nash equilibrium in the sparse-sampling($\epsilon$) based approximate delayed-VCG mechanism.*

**Proof:** Let $\tilde{\pi}$ denote the approximate policy computed by the sparse-sampling algorithm. Assume agents $\neq i$ are truthful. Now, if agent $i$ bids truthfully its expected utility is

$$E_{\theta|\theta_{\leq a_i}}\{v_i(\theta;\tilde{\pi}) + \sum_{j\neq i} R^j_{\leq T}(\theta;\tilde{\pi}) - \sum_{j\neq i} R^j_{\leq T}(\theta_{-i};\tilde{\pi})\} \qquad (4)$$

where the expectation is over both the types yet to be reported and the randomness introduced by the sparse-sampling($\epsilon$) algorithm. Substituting $R_{<a_i}(\theta_{<a_i};\tilde{\pi}) + V^{ss}(h_{a_i}(\theta_{\leq a_i};\tilde{\pi}))$ for the first two terms in Equation (4) and $R_{<a_i}(\theta_{<a_i};\tilde{\pi}) + V^{ss}(h_{a_i}(\theta_{\leq a_i\setminus i};\tilde{\pi}))$ for the third term, then its expected utility is at least

$$V^*(h_{a_i}(\theta_{\leq a_i};\tilde{\pi})) - V^{ss}(h_{a_i}(\theta_{\leq a_i\setminus i};\tilde{\pi})) - \epsilon \qquad (5)$$

because $V^{ss}(h_{a_i}(\theta_{\leq a_i};\tilde{\pi})) \geq V^*(h_{a_i}(\theta_{\leq a_i};\tilde{\pi})) - \epsilon$ by Lemma 1. Now, ignore term $R_{\leq T}(\theta_{-i};\tilde{\pi})$ in Equation (4), which is independent of agent $i$'s bid $\hat{\theta}_i$, and consider the maximal expected utility to agent $i$ from some non-truthful bid. The effect of $\hat{\theta}_i$ on the first two terms is indirect, through a change in the policy for periods $\geq a_i$. An agent can change the policy only indirectly, by changing the center's view of the state by misreporting its type. By definition, the agent can do no better than selecting optimal policy $\pi^*$, which is defined to maximize the expected value of the first two terms. Putting this together, the expected utility from $\hat{\theta}_i$ is at most $V^*(h_{a_i}(\theta_{\leq a_i};\tilde{\pi})) - V^{ss}(h_{a_i}(\theta_{\leq a_i\setminus i};\tilde{\pi}))$ and at most $\epsilon$ better than that from bidding truthfully. ∎

**Theorem 2** *Truthful bidding is an $4\epsilon$-Bayesian-Nash equilibrium in the sparse-sampling($\epsilon$) based approximate online VCG mechanism.*

**Proof:** Assume agents $\neq i$ bid truthfully, and consider report $\hat{\theta}_i$. Clearly, the policy implemented in the approximate online-VCG mechanism is the same as in the delayed-VCG mechanism for all $\hat{\theta}_i$. Left to show is that the expected value of the payments are within $3\epsilon$ for all $\hat{\theta}_i$. From this we conclude that the expected utility to agent $i$ in the approximate-VCG mechanism is always within $3\epsilon$ of that in the approximate delayed-VCG mechanism, and therefore $4\epsilon$-BNE by Lemma 2. The expected payment in the approximate online VCG mechanism is

$$E_{\theta|\theta_{\leq a_i}}\{R^i_{\leq T}(\hat{\theta};\tilde{\pi})\} - \left[ E\{\hat{V}^{ss}(h_{\hat{a}_i}(\hat{\theta}_{\leq \hat{a}_i};\tilde{\pi})\} - E\{\hat{V}^{ss}(h_{\hat{a}_i}(\hat{\theta}_{\leq \hat{a}_i\setminus i};\tilde{\pi})\} \right]$$

The value function computed by the sparse-sampling($\epsilon$) algorithm is a random variable to agent $i$ at the time of bidding, and the second and third expectations are over the randomness introduced by the sparse-sampling($\epsilon$) algorithm. The first term is the same as in the payment in the approximate delayed-VCG mechanism. By Corollary 1, the value function estimated in the sparse-sampling($\epsilon$) is near-optimal in expectation and the total of the second two terms is *at least* $V^*(h_{\hat{a}_i}(\hat{\theta}_{\leq \hat{a}_i\setminus i};\pi^*)) - V^*(h_{\hat{a}_i}(\hat{\theta}_{\leq \hat{a}_i};\pi^*)) - 2\epsilon$. Ignoring the first term in $p_i^{\text{Dvcg}}$, the expected payment in the approximate delayed-VCG mechanism is *no more* than $V^*(h_{\hat{a}_i}(\hat{\theta}_{\leq \hat{a}_i\setminus i};\pi^*)) - (V^*(h_{\hat{a}_i}(\hat{\theta}_{\leq \hat{a}_i};\pi^*)) - \epsilon)$ because of the near-optimality of the value function of the stochastic policy (Lemma 1). Putting this together, we have a maximum difference in expected payments of $3\epsilon$. Similar analysis yields a maximum difference of $3\epsilon$ when an upper-bound is taken on the payment in the online VCG mechanism and compared with a lower-bound on the payment in the delayed mechanism. ∎

**Theorem 3** *For any parameter $\epsilon > 0$, the sparse-sampling($\epsilon$) based approximate online VCG mechanism has $\epsilon$-efficiency in an $4\epsilon$-BNE.*

# 5   Empirical Evaluation of Approximate Online VCG

**The WiFi Problem.** The WiFi problem considers a fixed number of channels $C$ with a random number of agents (max $A$) that can arrive per period. The time horizon

$T = 50$. Agents demand a single channel and arrive with per-unit value, distributed i.i.d. $V = \{v_1, \ldots, v_k\}$ and duration in the system, distributed i.i.d. $D = \{d_1, \ldots, d_l\}$. The value model requires that any allocation to agent $i$ must be for contiguous periods, and be made while the agent is present (i.e., during periods $[t, a_i + d_i]$, for arrival $a_i$ and duration $d_i$). An agent's value for an allocation of duration $x$ is $v_i x$ where $v_i$ is its per-unit value. Let $d_{\max}$ denote the maximal possible allocated duration. We define the following MDP components:

**State space:** We use the following compact, sufficient, statistic of history: a *resource schedule* is a (weakly non-decreasing) vector of length $d_{\max}$ that counts the number of channels available in the current period and next $d_{\max} - 1$ periods given previous actions ($C$ channels are available after this); an *agent vector* of size $(k \times l)$ that provides a count of the number of agents present but not allocated for each possible per-unit value and each possible duration (the duration is automatically decremented when an agent remains in the system for a period without receiving an allocation); the time remaining until horizon $T$.

**Action space:** The policy can postpone an agent allocation, or allocate an agent to a channel for the remaining duration of the agent's time in the system if a channel is available, and the remaining duration is not greater than $d_{\max}$.

**Payoff function:** The reward at a time step is the summed value obtained from all agents for which an allocation is made in this time step. This is the *total* value such an agent will receive before it departs.

**Transition probabilities:** The change in resource schedule, and in the agent vector that relates to agents currently present, is deterministic. The random new additions to the agent vector at each step are unaffected by the actions and also independent of time.

**Mechanisms.** In the exact online VCG mechanism we compute an optimal policy, and optimal MDP values, offline using finite-horizon value iteration [7]. In the sparse-sampling($\epsilon$) approach, we define a sampling tree depth $L$ (perhaps $< T$) and sample each state $m$ times. This limited sampling depth places a lower-bound on the best possible approximation accuracy from the sparse-sampling algorithm. We also employ *agent pruning*, with the agent vector in the state representation pruned to remove dominated agents: consider agent type with duration $d$ and value $v$ and remove all but $C - N$ agents where $N$ is the number of agents that either have remaining duration $\leq d$ and value $> v$ or duration $< d$ and value $\geq v$. In computing payments we use *factoring*, and only determine VCG payments once for each type of agent to arrive. We compare performance with a simple *fixed-price allocation scheme* that given a particular problem, computes off-line a fixed number of periods and a fixed price (agents are queued and offered the price at random as resources become available) that yields the maximum expected total value.

**Results** In the default model, we set $C = 5$, $A = 5$, define the set of values $V = \{1, 2, 3\}$, define the set of durations $D = \{1, 2, 6\}$, with lookahead $L = 4$ and sampling width $m = 6$. All results are averaged over at least 10 instances, and experiments were performed on a 3GHz P4, with 512 MB RAM. Value and revenue is normalized by the total value demanded by all agents, i.e. the value with infinite capacity.[2] Looking first at economic properties, Figure 1(A) shows the effect of varying the number of agents from 2 to 12, comparing the value and revenue between the approximate online VCG mechanism and the fixed price mechanism. Notice that the MDP method dominates the price-based scheme for value, with a notable performance improvement over fixed price when demand is neither very low (no contention) nor very high (lots of competition). Revenue is also generally better from the MDP-based mechanism than in the fixed price scheme. Fig. 1(B) shows the similar effect of varying the number of channels from 3 to 10.

Turning now to computational properties, Figure 1 (C) illustrates the effectiveness of sparse-sampling, and also agent pruning, sampled over 100 instances. The model is very

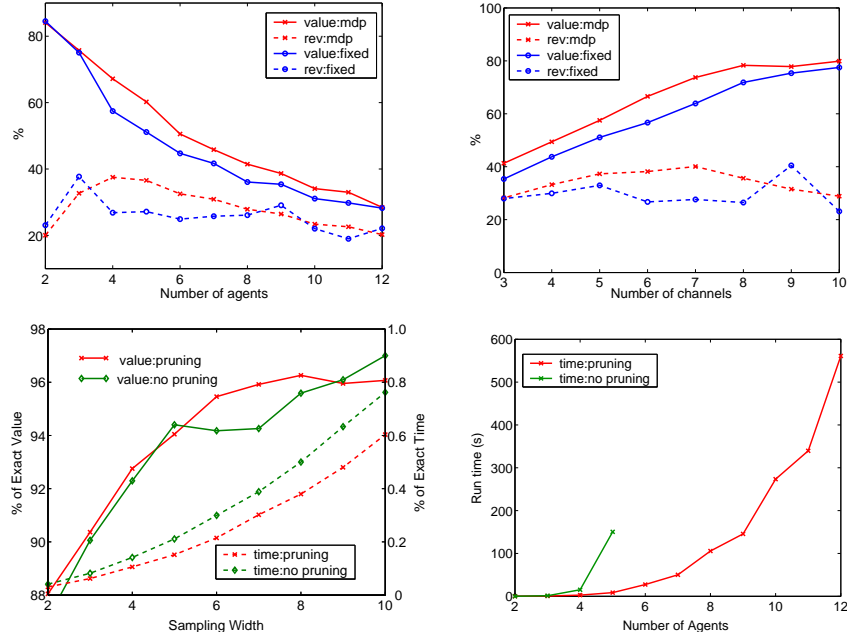

Figure 1: (A) Value and Revenue vs. Number of Agents. (B) Value and Revenue vs. Number of Channels. (C) Effect of Sampling Width. (D) Pruning speed-up.

small: $A = 2, C = 2, D = \{1, 2, 3\}, V = \{1, 2, 3\}$ and $L = 4$, to allow a comparison with the compute time for an optimal policy. The sparse-sampling method is already running in less than 1% of the time for optimal value-iteration (right-hand axis), with an accuracy as high as 96% of the optimal. Pruning provides an incremental speed-up, and actually improves accuracy, presumably by making better use of each sample. Figure 1 (D) shows that pruning is extremely useful computationally (in comparison with plain sparse-sampling), for the default model parameters and as the number of agents is increased from 2 to 12. Pruning is effective, removing around 50% of agents (summed across all states in the lookahead tree) at 5 agents.

**Acknowledgments.** David Parkes was funded by NSF grant IIS-0238147. Satinder Singh was funded by NSF grant CCF 0432027 and by a grant from DARPA's IPTO program.

## Footnotes

[1] Using histories as state will make the state space very large. Often, there will be some function $g$ for which $g(h)$ is a sufficient statistic for all possible states $h$. We ignore this possibility here.

[2]This explains why the value appears to drop as we scale up the number of agents— the total available value is increasing but supply remains fixed.

# References

[1] Eric Friedman and David C. Parkes. Pricing WiFi at Starbucks– Issues in online mechanism design. In *Fourth ACM Conf. on Electronic Commerce (EC'03)*, pages 240–241, 2003.

[2] Matthew O. Jackson. Mechanism theory. In *The Encyclopedia of Life Support Systems*. EOLSS Publishers, 2000.

[3] Michael Kearns, Yishay Mansour, and Andrew Y Ng. A sparse sampling algorithm for near-optimal planning in large Markov Decision Processes. In *Proc. 16th Int. Joint Conf. on Artificial Intelligence*, pages 1324–1331, 1999. To appear in Special Issue of *Machine Learning*.

[4] Ron Lavi and Noam Nisan. Competitive analysis of incentive compatible on-line auctions. In *Proc. 2nd ACM Conf. on Electronic Commerce (EC-00)*, 2000.

[5] Martin J Osborne and Ariel Rubinstein. *A Course in Game Theory*. MIT Press, 1994.

[6] David C. Parkes and Satinder Singh. An MDP-based approach to Online Mechanism Design. In *Proc. 17th Annual Conf. on Neural Information Processing Systems (NIPS'03)*, 2003.

[7] M L Puterman. *Markov decision processes: Discrete stochastic dynamic programming*. John Wiley & Sons, New York, 1994.